# Fast, smooth and adaptive regression in metric spaces

**Samory Kpotufe**
UCSD CSE

## Abstract

It was recently shown that certain nonparametric regressors can escape the curse of dimensionality when the intrinsic dimension of data is low ([1, 2]). We prove some stronger results in more general settings. In particular, we consider a regressor which, by combining aspects of both tree-based regression and kernel regression, adapts to intrinsic dimension, operates on general metrics, yields a smooth function, and evaluates in time $O(\log n)$. We derive a tight convergence rate of the form $n^{-2/(2+d)}$ where $d$ is the Assouad dimension of the input space.

## 1 Introduction

Relative to parametric methods, nonparametric regressors require few structural assumptions on the function being learned. However, their performance tends to deteriorate as the number of features increases. This so-called curse of dimensionality is quantified by various lower bounds on the convergence rates of the form $n^{-2/(2+D)}$ for data in $\mathbb{R}^D$ (see e.g. [3, 4]). In other words, one might require a data size exponential in $D$ in order to attain a low risk.

Fortunately, it is often the case that data in $\mathbb{R}^D$ has low intrinsic complexity, e.g. the data is near a manifold or is sparse, and we hope to exploit such situations. One simple approach, termed *manifold learning* (e.g. [5, 6, 7]), is to embed the data into a lower dimensional space where the regressor might work well. A recent approach with theoretical guarantees for nonparametric regression, is the study of *adaptive* procedures, i.e. ones that operate in $\mathbb{R}^D$ but attain convergence rates that depend just on the intrinsic dimension of data. An initial result [1] shows that for data on a $d$-dimensional manifold, the asymptotic risk at a point $x \in \mathbb{R}^D$ depends just on $d$ and on the behavior of the distribution in a neighborhood of $x$. Later, [2] showed that a regressor based on the RPtree of [8] (a hierarchical partitioning procedure) is not only fast to evaluate, but is adaptive to *Assouad dimension*, a measure which captures notions such as manifold dimension and data sparsity. The related notion of *box dimension* (see e.g. [9]) was shown in an earlier work [10] to control the risk of nearest neighbor regression, although adaptivity was not a subject of that result.

This work extends the applicability of such adaptivity results to more general uses of nonparametric regression. In particular, we present an adaptive regressor which, unlike RPtree, operates on a general metric space where only distances are provided, and yields a smooth function, an important property in many domains (see e.g. [11] which considers the smooth control of a robotic tool based on noisy outside input). In addition, our regressor can be evaluated in time just $O(\log n)$, unlike kernel or nearest neighbor regression. The evaluation time for these two forms of regression is lower bounded by the number of sample points contributing to the regression estimate. For nearest neighbor regression, this number is given by a parameter $k_n$ whose optimal setting (see [12]) is $O\left(n^{2/(2+d)}\right)$. For kernel regression, given an optimal bandwidth $h \approx n^{-1/(2+d)}$ (see [12]), we would expect about $nh^d \approx n^{2/(2+d)}$ points in the ball $B(x, h)$ around a query point $x$.

We note that there exist many heuristics for speeding up kernel regression, which generally combine fast proximity search procedures with other elaborate methods for approximating the kernel weights (see e.g. [13, 14, 15]). There are no rigorous bounds on either the achievable speedup or the risk of the resulting regressor.

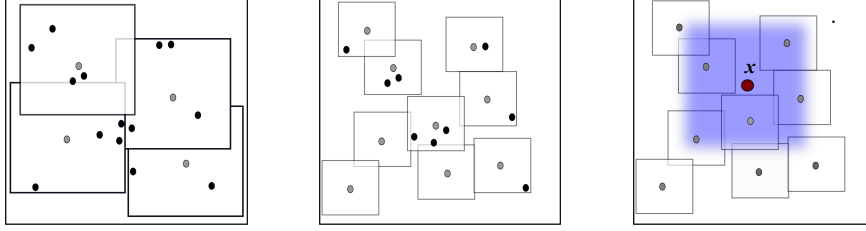

Figure 1: *Left and Middle-* Two $r$-nets at different scales $r$, each net inducing a partition of the sample **X**. In each case, the gray points are the $r$-net centers. For regression each center contributes the average $Y$ value of the data points assigned to them (points in the cells). *Right-* Given an $r$-net and a bandwidth $h$, a kernel around a query point $x$ weights the $Y$-contribution of each center to the regression estimate for $x$.

Our regressor integrates aspects of both tree-based regression and kernel regression. It constructs partitions of the input dataset $\mathbf{X} = \{X_i\}_1^n$, and uses a kernel to select a few sets within a given partition, each set contributing its average output $Y$ value to the estimate. We show that such a regressor achieves an excess risk of $O\left(n^{-2/(2+d)}\right)$, where $d$ is the Assouad dimension of the input data space. This is a tighter convergence rate than the $O\left(n^{-2/(2+O(d\log d))}\right)$ of RPtree regression (see [2]). Finally, the evaluation time of $O(\log n)$ is arrived at by modifying the *cover tree* proximity search procedure of [16]. Unlike in [16], this guarantee requires no growth assumption on the data distribution.

We'll now proceed with a more detailed presentation of the results in the next section, followed by technical details in sections 3 and 4.

## 2 Detailed overview of results

We're given i.i.d training data $(\mathbf{X}, \mathbf{Y}) = \{(X_i, Y_i)\}_1^n$, where the input variable $X$ belongs to a metric space $\mathcal{X}$ where the distance between points is given by the metric $\rho$, and the output $Y$ belongs to a subset $\mathcal{Y}$ of some Euclidean space. We'll let $\Delta_{\mathcal{X}}$ and $\Delta_{\mathcal{Y}}$ denote the diameters of $\mathcal{X}$ and $\mathcal{Y}$.

**Assouad dimension:** The Assouad or *doubling* dimension of $\mathcal{X}$ is defined as the smallest $d$ such that any ball can be covered by $2^d$ balls of half its radius.

*Examples:* A $d$-dimensional affine subspace of a Euclidean space $\mathbb{R}^D$ has Assouad dimension $O(d)$ [9]. A $d$-dimensional submanifold of a Euclidean space $\mathbb{R}^D$ has Assouad dimension $O(d)$ subject to a bound on its curvature [8]. A $d$-sparse data space in $\mathbb{R}^D$, i.e. one where each data point has at most $d$ non zero coordinates, has Assouad dimension $O(d\log D)$ [8, 2].

The algorithm has no knowledge of the dimension $d$, nor of $\Delta_{\mathcal{Y}}$, although we assume $\Delta_{\mathcal{X}}$ is known (or can be upper-bounded).

**Regression function:** We assume the regression function $f(x) \doteq \mathbb{E}[Y|X = x]$ is Lipschitz, i.e. there exists $\lambda$, unknown, such that $\forall x, x' \in \mathcal{X}, \|f(x) - f(x')\| \leq \lambda \cdot \rho(x, x')$.

**Excess risk:** Our performance criteria for a regressor $f_n(x)$ is the integrated excess $l_2$ risk:

$$\|f_n - f\|^2 \doteq \underset{X}{\mathbb{E}}\|f_n(X) - f(X)\|^2 = \underset{X,Y}{\mathbb{E}}\|f_n(X) - Y\|^2 - \underset{X,Y}{\mathbb{E}}\|f(X) - Y\|^2. \quad (1)$$

### 2.1 Algorithm overview

We'll consider a set of partitions of the data induced by a hierarchy of $r$-nets of **X**. Here an $r$-net $\mathbf{Q}_r$ is understood to be both an $r$-cover of **X** (all points in **X** are within $r$ of some point in $\mathbf{Q}_r$), and an $r$-packing (the points in $\mathbf{Q}_r$ are at least $r$ apart). The details on how to build the $r$-nets are covered in section 4. For now, we'll consider a class of regressors defined over these nets (as illustrated in Figure 1), and we'll describe how to select a good regressor out of this class.

**Partitions of X:** The $r$-nets are denoted by $\left\{\mathbf{Q}_r, r \in \{\Delta_{\mathcal{X}}/2^i\}_0^{I+2}\right\}$, where $I \doteq \lceil \log n \rceil$, and $\mathbf{Q}_r \subset \mathbf{X}$. Each $\mathbf{Q} \in \left\{\mathbf{Q}_r, r \in \{\Delta_{\mathcal{X}}/2^i\}_0^{I+2}\right\}$ induces a partition $\{\mathbf{X}(q), q \in \mathbf{Q}\}$ of **X**, where

$\mathbf{X}(q)$ designate all those points in $\mathbf{X}$ whose closest point in $\mathbf{Q}$ is $q$. We set $n_q \doteq |\mathbf{X}(q)|$, and $\bar{Y}_q = \frac{1}{n_q} \sum_{i:X_i \in \mathbf{X}(q)} Y_i$.

**Admissible kernels:** We assume that $K(u)$ is a non increasing function of $u \in [0, \infty)$; $K$ is positive on $u \in [0, 1)$, maximal at $u = 0$, and vanishes for $u \geq 1$. To simplify notation, we'll often let $K(x, q, h)$ denote $K(\rho(x, q)/h)$.

**Regressors:** For each $\mathbf{Q} \in \left\{ \mathbf{Q}_r, r \in \{\Delta_{\mathcal{X}}/2^i\}_0^{I+2} \right\}$, and given a bandwidth $h$, we define the following regressor:

$$f_{n,\mathbf{Q}}(x) = \sum_{q \in \mathbf{Q}} w_q(x) \bar{Y}_q, \text{ where } w_q = \frac{n_q(K(x, q, h) + \epsilon)}{\sum_{q' \in \mathbf{Q}} n_{q'}(K(x, q', h) + \epsilon)}. \quad (2)$$

The positive constant $\epsilon$ ensures that the estimate remains well defined when $K(x, q, h) = 0$. We assume $\epsilon \leq K(1/2)/n^2$. We can view $(K(\cdot) + \epsilon)$ as the effective kernel which never vanishes. It is clear that the learned function $f_{n,\mathbf{Q}}$ inherits any degree of smoothness from the kernel function $K$, i.e. if $K$ is of class $C^k$, then so is $f_{n,\mathbf{Q}}$.

**Selecting the final regressor:** For fixed $n$, $K(\cdot)$, and $\{\mathbf{Q}_r, r \in \{\Delta_{\mathcal{X}}/2^i\}_0^{I+2}\}$, equation (2) above defines a class of regressors parameterized by $r \in \{\Delta_{\mathcal{X}}/2^i\}_0^{I+2}$, and the bandwidth $h$. We want to pick a good regressor out of this class. We can reduce the search space right away by noticing that we need $r = \theta(h)$: if $r \gg h$ then $B(x, h) \cap \mathbf{Q}_r$ is empty for most $x$ since the points in $\mathbf{Q}_r$ are over $r$ apart, and if $r \ll h$ then $B(x, h) \cap \mathbf{Q}_r$ might contain a lot of points, thus increasing evaluation time. So for each choice of $h$, we will set $r = h/4$, which will yield good guarantees on computational and prediction performance. The final regressor is selected as follows.

Draw a new sample $(\mathbf{X}', \mathbf{Y}')$ of size $n$. As before let $I \doteq \lceil \log n \rceil$, and define $H \doteq \{\Delta_{\mathcal{X}}/2^i\}_0^I$. For every $h \in H$, pick the r-net $\mathbf{Q}_{h/4}$ and test $f_{n,\mathbf{Q}_{h/4}}$ on $(\mathbf{X}', \mathbf{Y}')$; let the empirical risk be minimized at $h_o$, i.e. $h_o \doteq \operatorname{argmin}_{h \in H} \frac{1}{n} \sum_{i=1}^n \left\| f_{n,\mathbf{Q}_{h/4}}(X_i') - Y_i' \right\|^2$. Return $f_{n,\mathbf{Q}_{h_o/4}}$ as the final regressor.

**Fast evaluation:** Each regressor $f_{n,\mathbf{Q}_{h/4}}(x)$ can be estimated quickly on points $x$ by traversing (nested) $r$-nets as described in detail in section 4.

## 2.2 Computational and prediction performance

The cover property ensures that for some $h$, $\mathbf{Q}_{h/4}$ is a good summary of local information (for prediction performance), while the packing property ensures that few points in $\mathbf{Q}_{h/4}$ fall in $B(x, h)$ (for fast evaluation). We have the following main result.

**Theorem 1.** *Let $d$ be the Assouad dimension of $\mathcal{X}$ and let $n \geq \max\left( 9, \left(\frac{\Delta_{\mathcal{Y}}}{\lambda \Delta_{\mathcal{X}}}\right)^2, \left(\frac{\lambda \Delta_{\mathcal{X}}}{\Delta_{\mathcal{Y}}}\right)^2 \right)$.*

*(a) The final regressor selected satisfies*

$$\mathbb{E} \left\| f_{n,\mathbf{Q}_{h_o/4}} - f \right\|^2 \leq C \left(\lambda \Delta_{\mathcal{X}}\right)^{2d/(2+d)} \left(\frac{\Delta_{\mathcal{Y}}^2}{n}\right)^{2/(2+d)} + 3\Delta_{\mathcal{Y}}^2 \sqrt{\frac{\ln(n \log n)}{n}},$$

*where $C$ depends on the Assouad dimension $d$ and on $K(0)/K(1/2)$.*

*(b) $f_{n,\mathbf{Q}_{h_o/4}}(x)$ can be computed in time $C' \log n$, where $C'$ depends just on $d$.*

Part (a) of Theorem 1 is given by Corollary 1 of section 3, and does not depend on how the $r$-nets are built; part (b) follows from Lemma 4 of section 4 which specifies the nets.

## 3 Risk analysis

Throughout this section we assume $0 < h < \Delta_{\mathcal{X}}$ and we let $\mathbf{Q} = \mathbf{Q}_{h/4}$. We'll bound the risk for $f_{n,\mathbf{Q}}$ for any fixed choice of $h$, and then show that the final $h_0$ selected yields a good risk. The results in this section only require the fact that $\mathbf{Q}$ is a cover of data and thus preserves local information, while the packing property is needed in the next section for fast evaluation.

Define $\widetilde{f}_{n,\mathbf{Q}}(x) \doteq \mathbb{E}_{\mathbf{Y}|\mathbf{X}} f_{n,\mathbf{Q}}(x)$, i.e. the conditional expectation of the estimate, for $\mathbf{X}$ fixed. We have the following standard decomposition of the excess risk into variance and bias terms:

$$\forall x \in \mathcal{X}, \ \underset{\mathbf{Y}|\mathbf{X}}{\mathbb{E}} \|f_{n,\mathbf{Q}}(x) - f(x)\|^2 = \underset{\mathbf{Y}|\mathbf{X}}{\mathbb{E}} \left\| f_{n,\mathbf{Q}}(x) - \widetilde{f}_{n,\mathbf{Q}}(x) \right\|^2 + \left\| \widetilde{f}_{n,\mathbf{Q}}(x) - f(x) \right\|^2. \quad (3)$$

We'll proceed by bounding each term separately in the following two lemmas, and then combining these bounds in Lemma 3. We'll let $\mu$ denote the marginal measure over $\mathcal{X}$ and $\mu_n$ denote the corresponding empirical measure.

**Lemma 1** (Variance at $x$). *Fix* $\mathbf{X}$*, and let* $\mathbf{Q}$ *be an* $\frac{h}{4}$*-net of* $\mathbf{X}$*,* $0 < h < \Delta_{\mathcal{X}}$*. Consider* $x \in \mathcal{X}$ *such that* $\mathbf{X} \cap (B(x, h/4)) \neq \emptyset$*. We have*

$$\underset{\mathbf{Y}|\mathbf{X}}{\mathbb{E}} \left\| f_{n,\mathbf{Q}}(x) - \widetilde{f}_{n,\mathbf{Q}}(x) \right\|^2 \leq \frac{2K(0)\Delta_{\mathcal{Y}}^2}{K(1/2) \cdot n\mu_n \left( B(x, h/4) \right)}.$$

*Proof.* Remember that for independent random vectors $v_i$ with expectation $\mathbf{0}$, $\mathbb{E} \|\sum_i v_i\|^2 = \sum_i \mathbb{E} \|v_i\|^2$. We apply this fact twice in the inequalities below, given that, conditioned on $\mathbf{X}$ and $\mathbf{Q} \subset \mathbf{X}$, the $Y_i$ values are mutually independent and so are the $\bar{Y}_q$ values. We have

$$
\begin{aligned}
\underset{\mathbf{Y}|\mathbf{X}}{\mathbb{E}} \left\| f_{n,\mathbf{Q}}(x) - \widetilde{f}_{n,\mathbf{Q}}(x) \right\|^2 &= \underset{\mathbf{Y}|\mathbf{X}}{\mathbb{E}} \left\| \sum_{q \in \mathbf{Q}} w_q(x) \left( \bar{Y}_q - \underset{\mathbf{Y}|\mathbf{X}}{\mathbb{E}} \bar{Y}_q \right) \right\|^2 \leq \sum_{q \in \mathbf{Q}} w_q^2(x) \underset{\mathbf{Y}|\mathbf{X}}{\mathbb{E}} \left\| \bar{Y}_q - \underset{\mathbf{Y}|\mathbf{X}}{\mathbb{E}} \bar{Y}_q \right\|^2 \\
&= \sum_{q \in \mathbf{Q}} w_q^2(x) \underset{\mathbf{Y}|\mathbf{X}}{\mathbb{E}} \left\| \sum_{i: X_i \in \mathbf{X}(q)} \frac{1}{n_q} \left( Y_i - \underset{\mathbf{Y}|\mathbf{X}}{\mathbb{E}} Y_i \right) \right\|^2 \leq \sum_{q \in \mathbf{Q}} w_q^2(x) \frac{\Delta_{\mathcal{Y}}^2}{n_q} \\
&\leq \left( \max_{q \in \mathbf{Q}} \left\{ w_q(x) \frac{\Delta_{\mathcal{Y}}^2}{n_q} \right\} \right) \sum_{q \in \mathbf{Q}} w_q = \max_{q \in \mathbf{Q}} \left\{ w_q(x) \frac{\Delta_{\mathcal{Y}}^2}{n_q} \right\} \\
&= \max_{q \in \mathbf{Q}} \frac{(K(x, q, h) + \epsilon) \Delta_{\mathcal{Y}}^2}{\sum_{q' \in \mathbf{Q}} n_{q'} (K(x, q', h) + \epsilon)} \leq \frac{2K(0)\Delta_{\mathcal{Y}}^2}{\sum_{q \in \mathbf{Q}} n_q K(x, q, h)}. \quad (4)
\end{aligned}
$$

To bound the fraction in (4), we lower-bound the denominator as:

$$\sum_{q \in \mathbf{Q}} n_q K(x, q, h) \geq \sum_{q: \rho(x,q) \leq h/2} n_q K(x, q, h) \geq \sum_{q: \rho(x,q) \leq h/2} n_q K(1/2) \geq K(1/2) \cdot n\mu_n(B(x, h/4)).$$

The last inequality follows by remarking that, since $\mathbf{Q}$ is an $\frac{h}{4}$-cover of $\mathbf{X}$, the ball $B(x, h/4)$ can only contain points from $\cup_{q: \rho(x,q) \leq h/2} \mathbf{X}(q)$. Plug this last inequality into (4) and conclude. $\square$

**Lemma 2** (Bias at $x$). *As before, fix* $\mathbf{X}$*, and let* $\mathbf{Q}$ *be an* $\frac{h}{4}$*-net of* $\mathbf{X}$*,* $0 < h < \Delta_{\mathcal{X}}$*. Consider* $x \in \mathcal{X}$ *such that* $\mathbf{X} \cap (B(x, h/4)) \neq \emptyset$*. We have*

$$\left\| \widetilde{f}_{n,\mathbf{Q}}(x) - f(x) \right\|^2 \leq 2\lambda^2 h^2 + \frac{\Delta_{\mathcal{Y}}^2}{n}.$$

*Proof.* We have

$$\left\| \widetilde{f}_{n,\mathbf{Q}}(x) - f(x) \right\|^2 = \left\| \sum_{q \in \mathbf{Q}} \frac{w_q(x)}{n_q} \sum_{X_i \in \mathbf{X}(q)} (f(X_i) - f(x)) \right\|^2 \leq \sum_{q \in \mathbf{Q}} \frac{w_q(x)}{n_q} \sum_{X_i \in \mathbf{X}(q)} \|f(X_i) - f(x)\|^2,$$

where we just applied Jensen's inequality on the norm square. We bound the r.h.s by breaking the summation over two subsets of $\mathbf{Q}$ as follows.

$$
\begin{aligned}
\sum_{q: \rho(x,q) < h} \frac{w_q(x)}{n_q} \sum_{X_i \in \mathbf{X}(q)} \|f(X_i) - f(x)\|^2 &\leq \sum_{q: \rho(x,q) < h} \frac{w_q(x)}{n_q} \sum_{X_i \in \mathbf{X}(q)} \lambda^2 \rho(X_i, x)^2 \\
\leq \sum_{q: \rho(x,q) < h} \frac{w_q(x)}{n_q} \sum_{X_i \in \mathbf{X}(q)} \lambda^2 (\rho(x, q) + \rho(q, X_i))^2 &\leq \sum_{q: \rho(x,q) < h} \frac{w_q(x)}{n_q} \sum_{X_i \in \mathbf{X}(q)} \frac{25}{16} \lambda^2 h^2 \leq 2\lambda^2 h^2.
\end{aligned}
$$

Next, we have

$$\sum_{q:\rho(x,q)\geq h} \frac{w_q(x)}{n_q} \sum_{X_i\in\mathbf{X}(q)} \|f(X_i)-f(x)\|^2 \leq \sum_{q:\rho(x,q)\geq h} w_q(x)\Delta_{\mathcal{Y}}^2$$

$$= \frac{\Delta_{\mathcal{Y}}^2 \sum_{q:\rho(x,q)\geq h} n_q\epsilon}{\displaystyle\sum_{q:\rho(x,q)\geq h} n_q\epsilon + \sum_{q:\rho(x,q)<h} n_q\left(K(x,q,h)+\epsilon\right)} = \Delta_{\mathcal{Y}}^2 \left(1 + \frac{\sum_{q:\rho(x,q)<h} n_q\left(K(x,q,h)+\epsilon\right)}{\displaystyle\sum_{q:\rho(x,q)\geq h} n_q\epsilon}\right)^{-1}$$

$$\leq \Delta_{\mathcal{Y}}^2 \left(1 + \frac{K(1/2)}{\sum_{q:\rho(x,q)\geq h} n_q\epsilon}\right)^{-1} \leq \Delta_{\mathcal{Y}}^2 \left(1 + \frac{K(1/2)}{n\epsilon}\right)^{-1} \leq \frac{\Delta_{\mathcal{Y}}^2}{1+n},$$

where the second inequality is due to the fact that, since $\mu_n(B(x,h/4))>0$, the set $B(x,h/2)\cap\mathbf{Q}$ cannot be empty (remember that $\mathbf{Q}$ is an $\frac{h}{4}$-cover of $\mathbf{X}$). This concludes the argument. □

**Lemma 3** (Integrated excess risk). *Let $\mathbf{Q}$ be an $\frac{h}{4}$-net of $\mathbf{X}$, $0 < h < \Delta_{\mathcal{X}}$. We have*

$$\mathop{\mathbb{E}}_{(\mathbf{X},\mathbf{Y})} \|f_{n,\mathbf{Q}} - f\|^2 \leq C_0 \frac{\Delta_{\mathcal{Y}}^2}{n\cdot(h/\Delta_{\mathcal{X}})^d} + 2\lambda^2 h^2,$$

*where $C_0$ depends on the Assouad dimension $d$ and on $K(0)/K(1/2)$.*

*Proof.* Applying Fubini's theorem, the expected excess risk, $\mathbb{E}_{(\mathbf{X},\mathbf{Y})} \|f_{n,\mathbf{Q}} - f\|^2$, can be written as

$$\mathop{\mathbb{E}}_X \mathop{\mathbb{E}}_{(\mathbf{X},\mathbf{Y})} \|f_{n,\mathbf{Q}}(X) - f(X)\|^2 \left(\mathbb{1}_{\{\mu_n(B(X,h/4))>0\}} + \mathbb{1}_{\{\mu_n(B(X,h/4))=0\}}\right).$$

By lemmas 1 and 2 we have for $X = x$ fixed,

$$\mathop{\mathbb{E}}_{(\mathbf{X},\mathbf{Y})} \|f_{n,\mathbf{Q}}(x) - f(x)\|^2 \mathbb{1}_{\{\mu_n(B(x,h/4))>0\}} \leq C_1 \mathop{\mathbb{E}}_{\mathbf{X}} \left[\frac{\Delta_{\mathcal{Y}}^2 \mathbb{1}_{\{\mu_n(B(x,h/4))>0\}}}{n\mu_n(B(x,h/4))}\right] + 2\lambda^2 h^2 + \frac{\Delta_{\mathcal{Y}}^2}{n}$$

$$\leq C_1 \left(\frac{2\Delta_{\mathcal{Y}}^2}{n\mu(B(x,h/4))}\right) + 2\lambda^2 h^2 + \frac{\Delta_{\mathcal{Y}}^2}{n}, \quad (5)$$

where for the last inequality we used the fact that for a binomial $b(n,p)$, $\mathbb{E}\left[\frac{\mathbb{1}_{\{b(n,p)>0\}}}{b(n,p)}\right] \leq \frac{2}{np}$ (see lemma 4.1 of [12]).

For the case where $B(x,h/4)$ is empty, we have

$$\mathop{\mathbb{E}}_{(\mathbf{X},\mathbf{Y})} \|f_{n,\mathbf{Q}}(x) - f(x)\|^2 \mathbb{1}_{\{\mu_n(B(x,h/4))=0\}} \leq \Delta_{\mathcal{Y}}^2 \mathop{\mathbb{E}}_{\mathbf{X}} \mathbb{1}_{\{\mu_n(B(x,h/4))=0\}} = \Delta_{\mathcal{Y}}^2 \left(1 - \mu(B(x,h/4))\right)^n$$

$$\leq \Delta_{\mathcal{Y}}^2 e^{-n\mu(B(x,h/4))} \leq \frac{\Delta_{\mathcal{Y}}^2}{n\mu(B(x,h/4))}. \quad (6)$$

Combining (6) and (5), we can then bound the expected excess risk as

$$\mathop{\mathbb{E}}_{(\mathbf{X},\mathbf{Y})} \|f_{n,\mathbf{Q}} - f\|^2 \leq \frac{3C_1\Delta_{\mathcal{Y}}^2}{n} \mathop{\mathbb{E}}_X \left[\frac{1}{\mu(B(X,h/4))}\right] + 2\lambda^2 h^2 + \frac{\Delta_{\mathcal{Y}}^2}{n}. \quad (7)$$

The expectation on the r.h.s is bounded using a standard covering argument (see e.g. [12]). Let $\{z_i\}_1^N$ be an $\frac{h}{8}$-cover of $\mathcal{X}$. Notice that for any $z_i$, $x \in B(z_i,h/8)$ implies $B(x,h/4) \supset B(z_i,h/8)$. We therefore have

$$\mathop{\mathbb{E}}_X \left[\frac{1}{\mu(B(X,h/4))}\right] \leq \sum_{i=1}^N \mathop{\mathbb{E}}_X \left[\frac{\mathbb{1}_{\{X\in B(z_i,h/8)\}}}{\mu(B(X,h/4))}\right] \leq \sum_{i=1}^N \mathop{\mathbb{E}}_X \left[\frac{\mathbb{1}_{\{X\in B(z_i,h/8)\}}}{\mu(B(X,h/8))}\right]$$

$$= N \leq C_2 \left(\frac{\Delta_{\mathcal{X}}}{h}\right)^d, \text{ where } C_2 \text{ depends just on } d.$$

We conclude by combining the above with (7) to obtain

$$\mathop{\mathbb{E}}_{(\mathbf{X},\mathbf{Y})} \|f_{n,\mathbf{Q}} - f\|^2 \leq \frac{3C_1 C_2 \Delta_{\mathcal{Y}}^2}{n(h/\Delta_{\mathcal{X}})^d} + 2\lambda^2 h^2 + \frac{\Delta_{\mathcal{Y}}^2}{n}.$$

□

**Corollary 1.** *Let* $n \geq \max\left(9, \left(\frac{\Delta_{\mathcal{Y}}}{\lambda \Delta_{\mathcal{X}}}\right)^2, \left(\frac{\lambda \Delta_{\mathcal{X}}}{\Delta_{\mathcal{Y}}}\right)^2\right)$. *The final regressor selected satisfies*

$$\mathbb{E}\left\|f_{n,\mathbf{Q}_{h_o/4}} - f\right\|^2 \leq C\left(\lambda \Delta_{\mathcal{X}}\right)^{2d/(2+d)}\left(\frac{\Delta_{\mathcal{Y}}^2}{n}\right)^{2/(2+d)} + 3\Delta_{\mathcal{Y}}^2\sqrt{\frac{\ln(n\log n)}{n}},$$

*where $C$ depends on the Assouad dimension $d$ and on $K(0)/K(1/2)$.*

*Proof outline.* Let $\tilde{h} = C_3\left(\Delta_{\mathcal{X}}^{d/(2+d)}\left(\frac{\Delta_{\mathcal{Y}}^2}{\lambda^2 n}\right)^{1/(2+d)}\right) \in H$. We note that $n$ is lower bounded so that such an $\tilde{h}$ is in $H$. We have by Lemma 3 that for $\tilde{h}$,

$$\mathbb{E}_{\mathbf{X},\mathbf{Y}}\left\|f_{n,\mathbf{Q}_{\tilde{h}/4}} - f\right\|^2 \leq C_0\left(\lambda \Delta_{\mathcal{X}}\right)^{2d/(2+d)}\left(\frac{\Delta_{\mathcal{Y}}^2}{n}\right)^{2/(2+d)}.$$

Applying McDiarmid's to the empirical risk followed by a union bound over $H$, we have that, with probability at least $1 - 1/\sqrt{n}$ over the choice of $(\mathbf{X}', \mathbf{Y}')$, for all $h \in H$

$$\left|\mathbb{E}_{X,Y}\left\|f_{n,\mathbf{Q}_{h/4}}(X) - Y\right\|^2 - \frac{1}{n}\sum_{i=0}^{n}\left\|f_{n,\mathbf{Q}_{h/4}}(X_i') - Y_i'\right\|\right| \leq \Delta_{\mathcal{Y}}^2\sqrt{\frac{\ln(|H|\sqrt{n})}{n}}.$$

It follows that $\mathbb{E}_{X,Y}\left\|f_{n,\mathbf{Q}_{h_o/4}}(X) - Y\right\|^2 \leq \mathbb{E}_{X,Y}\left\|f_{n,\mathbf{Q}_{\tilde{h}/4}}(X) - Y\right\|^2 + 2\Delta_{\mathcal{Y}}^2\sqrt{\frac{\ln(|H|\sqrt{n})}{n}}$, which by (1) implies $\left\|f_{n,\mathbf{Q}_{h_o/4}} - f\right\|^2 \leq \left\|f_{n,\mathbf{Q}_{\tilde{h}/4}} - f\right\|^2 + 2\Delta_{\mathcal{Y}}^2\sqrt{\frac{\ln(|H|\sqrt{n})}{n}}$. Take the expectation (given the randomness in the two samples) over this last inequality and conclude. □

## 4 Fast evaluation

In this section we show how to modify the cover-tree procedure of [16] to enable fast evaluation of $f_{n,\mathbf{Q}_{h/4}}$ for any $h \in H \doteq \{\Delta_{\mathcal{X}}/2^i\}_1^I$, $I = \lceil \log n \rceil$.

The cover-tree performs proximity search by navigating a hierarchy of nested $r$-nets of $\mathbf{X}$. The navigating-nets of [17] implement the same basic idea. They require additional book-keeping to enable range queries of the form $\mathbf{X} \cap B(x, h)$, for a query point $x$. Here we need to perform range searches of the form $\mathbf{Q}_{h/4} \cap B(x, h)$ and our book-keeping will therefore be different from [17]. Note that, for each $h$ and $\mathbf{Q}_{h/4}$, one could use a generic range search procedure such as [17] with the data in $\mathbf{Q}_{h/4}$ as input, but this requires building a separate data structure for each $h$, which is expensive. We use a single data structure.

### 4.1 The hierarchy of nets

Consider an ordering $\left\{X_{(i)}\right\}_1^n$ of the data points obtained as follows: $X_{(1)}$ and $X_{(2)}$ are the farthest points in $\mathbf{X}$; inductively for $2 < i < n$, $X_{(i)}$ in $\mathbf{X}$ is the farthest point from $\left\{X_{(1)}, \ldots, X_{(i-1)}\right\}$, where the distance to a set is defined as the minimum distance to a point in the set.

For $r \in \left\{\Delta_{\mathcal{X}}/2^i\right\}_0^{I+2}$, define $\mathbf{Q}_r = \left\{X_{(1)}, \ldots, X_{(i)}\right\}$ where $i \geq 1$ is the highest index such that $\rho\left(X_{(i)}, \left\{X_{(1)}, \ldots, X_{(i-1)}\right\}\right) \geq r$. Notice that, by construction, $\mathbf{Q}_r$ is an $r$-net of $\mathbf{X}$.

### 4.2 Data structure

The data structure consists of an acyclic directed graph, and *range* sets defined below.

**Neighborhood graph:** The nodes of the graph are the $\left\{X_{(i)}\right\}_1^n$, and the edges are given by the following parent-child relationship: starting at $r = \Delta_{\mathcal{X}}/2$, the parent of each node in $\mathbf{Q}_r \setminus \mathbf{Q}_{2r}$ is the point it is closest to in $\mathbf{Q}_{2r}$. The graph is implemented by maintaining an ordered list of children for each node, where the order is given by the children's appearance in the sequence $\left\{X_{(i)}\right\}_1^n$. These relationships are depicted in Figure 2.

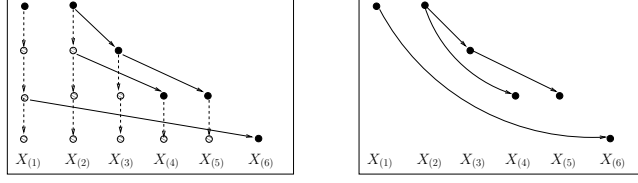

Figure 2: The $r$-nets (rows of left subfigure) are implicit to an ordering of the data. They define a parent-child relationship implemented by the *neighborhood graph* (right), the structure traversed for fast evaluation.

These ordered lists of children are used to implement the operation `nextChildren` defined iteratively as follows. Given $\mathbf{Q} \subset \left\{X_{(i)}\right\}_1^n$, let *visited* children denote any child of $q \in \mathbf{Q}$ that a previous call to `nextChildren` has already returned. The call `nextChildren` $(\mathbf{Q})$ returns children of $q \in \mathbf{Q}$ that have not yet been visited, starting with the unvisited child with lowest index in $\left\{X_{(i)}\right\}_1^n$, say $X_{(i)}$, and returning all unvisited children in $\mathbf{Q}_r$, the first net containing $X_{(i)}$, i.e. $X_{(i)} \in \mathbf{Q}_r \setminus \mathbf{Q}_{2r}$ ; $r$ is also returned. The children returned are then marked off as visited. The time complexity of this routine is just the number of children returned.

**Range sets:** For each node $X_{(i)}$ and each $r \in \left\{\Delta_{\mathcal{X}}/2^i\right\}_0^\infty$, we maintain a set of neighbors of $X_{(i)}$ in $\mathbf{Q}_r$ defined as $\mathbf{R}_{(i),r} \doteq \left\{q \in \mathbf{Q}_r : \rho\left(X_{(i)}, q\right) \le 8r\right\}$.

## 4.3 Evaluation

---
**Procedure** `evaluate` $(x, h)$

---
$\mathbf{Q} \leftarrow \mathbf{Q}_{\Delta_{\mathcal{X}}}$;
**repeat**
    $\mathbf{Q}', r \leftarrow$ `nextChildren` $(\mathbf{Q})$;
    $\mathbf{Q}'' \leftarrow \mathbf{Q} \cup \mathbf{Q}'$;
    **if** $r < h/4$ *or* $\mathbf{Q}' = \emptyset$ **then** // We reached past $\mathbf{Q}_{h/4}$.

        $X_{(i)} \leftarrow \operatorname{argmin}_{q \in \mathbf{Q}} \rho(x, q)$; // Closest point to $x$ in $\mathbf{Q}_{h/4}$.
        $\mathbf{Q} \leftarrow \mathbf{R}_{(i),h/4} \cap B(x, h)$; // Search in a range of $2h$ around $X_{(i)}$.
        Break loop ;
    **if** $\rho(x, \mathbf{Q}'') \ge h + 2r$ **then** // The set $\mathbf{Q}_{h/4} \cap B(x, h)$ is empty.

        $\mathbf{Q} \leftarrow \emptyset$;
        Break loop ;
    $\mathbf{Q} \leftarrow \{q \in \mathbf{Q}'', \rho(x, q) < \rho(x, \mathbf{Q}'') + 2r\}$;
**until** ... ;
//At this point $\mathbf{Q} = \mathbf{Q}_{h/4} \cap B(x, h)$.
**return**

$$f_{n, \mathbf{Q}_{h/4}}(x) \leftarrow \frac{\sum_{q \in \mathbf{Q}} n_q(K(x, q, h) + \epsilon)\bar{Y}_q + \epsilon \left(\sum_{q \in \mathbf{Q}_{h/4}} n_q \bar{Y}_q - \sum_{q \in \mathbf{Q}} n_q \bar{Y}_q\right)}{\sum_{q \in \mathbf{Q}} n_q(K(x, q, h) + \epsilon) + \epsilon \left(n - \sum_{q \in \mathbf{Q}} n_q\right)};$$

---

The evaluation procedure consists of quickly identifying the closest point $X_{(i)}$ to $x$ in $\mathbf{Q}_{h/4}$ and then searching in the range of $X_{(i)}$ for the points in $\mathbf{Q}_{h/4} \cap B(x, h)$. The identification of $X_{(i)}$ is done by going down the levels of nested nets, and discarding those points (and their descendants) that are certain to be farther to $x$ than $X_{(i)}$ (we will argue that "$\rho(x, \mathbf{Q}'') + 2r$" is an upper-bound on $\rho\left(x, X_{(i)}\right)$). Also, if $x$ is far enough from all points at the current level (second if-clause), we can safely stop early because $B(x, h)$ is unlikely to contain points from $\mathbf{Q}_{h/4}$ (we'll see that points in $\mathbf{Q}_{h/4}$ are all within $2r$ of their ancestor at the current level).

**Lemma 4.** *The call to procedure* `evaluate` $(x, h)$ *correctly evaluates* $f_{n, \mathbf{Q}_{h/4}}(x)$ *and has time complexity* $C \log \left(\Delta_{\mathcal{X}}/h\right) + \log n$ *where $C$ is at most* $2^{8d}$.

*Proof.* We first show that the algorithm correctly returns $f_{n,\mathbf{Q}_{\alpha h}}(x)$, and we then argue its run time.

*Correctness of* `evaluate`. The procedure works by first finding the closest point to $x$ in $\mathbf{Q}_{h/4}$, say $X_{(i)}$, and then identifying all nodes in $\left(\mathbf{R}_{(i),h/4} \cap B(x,h)\right) = \left(\mathbf{Q}_{h/4} \cap B(x,h)\right)$ (see the first if-clause). We just have to show that this closest point $X_{(i)}$ is correctly identified.

We'll argue the following loop invariant $\mathcal{I}$: at the beginning of the loop, $X_{(i)}$ is either in $\mathbf{Q}'' = \mathbf{Q} \cup \mathbf{Q}'$ or is a descendant of a node in $\mathbf{Q}'$. Let's consider some iteration where $\mathcal{I}$ holds (it certainly does in the first iteration).

If the first if-clause is entered, then $\mathbf{Q}$ is contained in $\mathbf{Q}_{h/4}$ but $\mathbf{Q}'$ is not, so $X_{(i)}$ must be in $\mathbf{Q}$ and we correctly return.

Suppose the first if-clause is not entered. Now let $X_{(j)}$ be the ancestor in $\mathbf{Q}'$ of $X_{(i)}$ or let it be $X_{(i)}$ itself if it's in $\mathbf{Q}''$. Let $r$ be as defined in `evaluate`, we have $\rho\left(X_{(i)}, X_{(j)}\right) < \sum_{k=0}^{\infty} r/2^k = 2r$ by going down the parent-child relations. It follows that

$$\rho\left(x, \mathbf{Q}''\right) \leq \rho\left(x, X_{(j)}\right) \leq \rho\left(x, X_{(i)}\right) + \rho\left(X_{(i)}, X_{(j)}\right) < \rho\left(x, X_{(i)}\right) + 2r.$$

In other words, we have $\rho\left(x, X_{(i)}\right) > \rho\left(x, \mathbf{Q}''\right) - 2r$. Thus, if the second if-clause is entered, we necessarily have $\rho\left(x, X_{(i)}\right) > h$, i.e. $B(x,h) \cap \mathbf{Q}_{h/4} = \emptyset$ and we correctly return.

Now assume none of the if-clauses is entered. Let $X_{(j)} \in \mathbf{Q}''$ be any of the points removed from $\mathbf{Q}''$ to obtain the next $\mathbf{Q}$. Let $X_{(k)}$ be a child of $X_{(j)}$ that has not yet been visited, or a descendant of such a child. If neither such $X_{(j)}$ or $X_{(k)}$ is $X_{(i)}$ then, by definition, $\mathcal{I}$ must hold at the next iteration. We sure have $X_{(j)} \neq X_{(i)}$ since $\rho\left(x, X_{(j)}\right) \geq \rho\left(x, \mathbf{Q}''\right) + 2r \geq \rho\left(x, X_{(i)}\right) + 2r$. Now notice that, by the same argument as above, $\rho\left(X_{(j)}, X_{(k)}\right) < \sum_{k=0}^{\infty} r/2^k = 2r$. We thus have $\rho\left(x, X_{(k)}\right) > \rho\left(x, X_{(j)}\right) - 2r \geq \rho\left(x, X_{(i)}\right)$ so we know $X_{(j)} \neq X_{(i)}$.

*Runtime of* `evaluate`. Starting from $\mathbf{Q}_{\Delta_{\mathcal{X}}}$, a different net $\mathbf{Q}_r$ is reached at every iteration, and the loop stops when we reach past $\mathbf{Q}_{h/4}$. Therefore the loop is entered at most $\log\left(\Delta_{\mathcal{X}}/h/4\right)$ times. In each iteration, most of the work is done parsing through $\mathbf{Q}''$, besides time spent for the range search in the last iteration. So the total runtime is $O\left(\log\left(\Delta_{\mathcal{X}}/h/4\right) \cdot \max|\mathbf{Q}''|\right)$ + range search time. We just need to bound $\max|\mathbf{Q}''| \leq \max|\mathbf{Q}| + \max|\mathbf{Q}'|$ and the range search time.

The following fact (see e.g. Lemma 4.1 of [9]) will come in handy: consider $r_1$ and $r_2$ such that $r_1/r_2$ is a power of 2, and let $B \subset \mathcal{X}$ be a ball of radius $r_1$; since $\mathcal{X}$ has Assouad dimension $d$, the smallest $r_2$-cover of $B$ is of size at most $(r_1/r_2)^d$, and the largest $r_2$-packing of $B$ is of size at most $(r_1/r_2)^{2d}$. This is true for any metric space, and therefore holds for $\mathbf{X}$ which is of Assouad dimension at most $d$ by inclusion in $\mathcal{X}$.

Let $\mathbf{Q}' \subset \mathbf{Q}_r$ so that $\mathbf{Q} \subset \mathbf{Q}_{2r}$ at the beginning of some iteration. Let $q \in \mathbf{Q}$, the children of $q$ in $\mathbf{Q}'$ are not in $\mathbf{Q}_{2r}$ and therefore are all within $2r$ of $\mathbf{Q}$; since these children an $r$-packing of $B(q, 2r)$, there are at most $2^{2d}$ of them. Thus, $\max|\mathbf{Q}'| \leq 2^{2d}\max|\mathbf{Q}|$.

Initially $\mathbf{Q} = \mathbf{Q}_{\Delta_{\mathcal{X}}}$ so we have $|\mathbf{Q}| \leq 2^{2d}$ since $\mathbf{Q}_{\Delta_{\mathcal{X}}}$ is a $\Delta_{\mathcal{X}}$-packing of $\mathbf{X} \subset B\left(X_{(1)}, 2\Delta_{\mathcal{X}}\right)$. At the end of each iteration we have $\mathbf{Q} \subset B(x, \rho(x, \mathbf{Q}'') + 2r)$. Now $\rho(x, \mathbf{Q}'') \leq h + 2r \leq 4r + 2r$ since the if-clauses were not entered if we got to the end of the iteration. Thus, $\mathbf{Q}$ is an $r$-packing of $B(x, 8r)$, and therefore $\max|\mathbf{Q}| \leq 2^{8d}$.

To finish, the range search around $X_{(i)}$ takes time $\left|\mathbf{R}_{(i),h/4}\right| \leq 2^{8d}$ since $\mathbf{R}_{(i),h/4}$ is an $\frac{h}{4}$-packing of $B\left(X_{(i)}, 2h\right)$. $\qquad\square$

### Acknowledgements

This work was supported by the National Science Foundation (under grants IIS-0347646, IIS-0713540, and IIS-0812598) and by a fellowship from the Engineering Institute at the Los Alamos National Laboratory. Many thanks to the anonymous NIPS reviewers for the useful comments, and thanks to Sanjoy Dasgupta for advice on the presentation.

# References

[1] P. Bickel and B. Li. Local polynomial regression on unknown manifolds. *Tech. Re. Dep. of Stats. UC Berkley*, 2006.

[2] S. Kpotufe. Escaping the curse of dimensionality with a tree-based regressor. *COLT*, 2009.

[3] C. J. Stone. Optimal rates of convergence for non-parametric estimators. *Ann. Statist.*, 8:1348–1360, 1980.

[4] C. J. Stone. Optimal global rates of convergence for non-parametric estimators. *Ann. Statist.*, 10:1340–1353, 1982.

[5] S. Roweis and L. Saul. Nonlinear dimensionality reduction by locally linear embedding. *Science*, 290:2323–2326, 2000.

[6] M. Belkin and N. Niyogi. Laplacian eigenmaps for dimensionality reduction and data representation. *Neural Computation*, 15:1373–1396, 2003.

[7] J.B. TenenBaum, V. De Silva, and J. Langford. A global geometric framework for non-linear dimensionality reduction. *Science*, 290:2319–2323, 2000.

[8] S. Dasgupta and Y. Freund. Random projection trees and low dimensional manifolds. *STOC*, 2008.

[9] K. Clarkson. Nearest-neighbor searching and metric space dimensions. *Nearest-Neighbor Methods for Learning and Vision: Theory and Practice*, 2005.

[10] S. Kulkarni and S. Posner. Rates of convergence of nearest neighbor estimation under arbitrary sampling. *IEEE Transactions on Information Theory*, 41, 1995.

[11] S. Schaal and C. Atkeson. Robot Juggling: An Implementation of Memory-based Learning. *Control Systems Magazine, IEEE*, 1994.

[12] L. Gyorfi, M. Kohler, A. Krzyzak, and H. Walk. *A Distribution Free Theory of Nonparametric Regression*. Springer, New York, NY, 2002.

[13] D. Lee and A. Grey. Faster gaussian summation: Theory and experiment. *UAI*, 2006.

[14] D. Lee and A. Grey. Fast high-dimensional kernel summations using the monte carlo multipole method. *NIPS*, 2008.

[15] C. Atkeson, A. Moore, and S. Schaal. Locally weighted learning. *AI Review*, 1997.

[16] A. Beygelzimer, S. Kakade, and J. Langford. Cover trees for nearest neighbors. *ICML*, 2006.

[17] R. Krauthgamer and J. Lee. Navigating nets: Simple algorithms for proximity search. *SODA*, 2004.

